# Latent Dirichlet Allocation

**David M. Blei, Andrew Y. Ng** and **Michael I. Jordan**
University of California, Berkeley
Berkeley, CA 94720

## Abstract

We propose a generative model for text and other collections of discrete data that generalizes or improves on several previous models including naive Bayes/unigram, mixture of unigrams [6], and Hofmann's aspect model, also known as probabilistic latent semantic indexing (pLSI) [3]. In the context of text modeling, our model posits that each document is generated as a mixture of topics, where the continuous-valued mixture proportions are distributed as a latent Dirichlet random variable. Inference and learning are carried out efficiently via variational algorithms. We present empirical results on applications of this model to problems in text modeling, collaborative filtering, and text classification.

## 1 Introduction

Recent years have seen the development and successful application of several latent factor models for discrete data. One notable example, Hofmann's pLSI/aspect model [3], has received the attention of many researchers, and applications have emerged in text modeling [3], collaborative filtering [7], and link analysis [1]. In the context of text modeling, pLSI is a "bag-of-words" model in that it ignores the ordering of the words in a document. It performs dimensionality reduction, relating each document to a position in low-dimensional "topic" space. In this sense, it is analogous to PCA, except that it is explicitly designed for and works on discrete data.

A sometimes poorly-understood subtlety of pLSI is that, even though it is typically described as a generative model, its documents have no generative probabilistic semantics and are treated simply as a set of labels for the specific documents seen in the training set. Thus there is no natural way to pose questions such as "what is the probability of this previously unseen document?". Moreover, since each training document is treated as a separate entity, the pLSI model has a large number of parameters and heuristic "tempering" methods are needed to prevent overfitting.

In this paper we describe a new model for collections of discrete data that provides full generative probabilistic semantics for documents. Documents are modeled via a hidden Dirichlet random variable that specifies a probability distribution on a latent, low-dimensional topic space. The distribution over words of an unseen document is a continuous mixture over document space and a discrete mixture over all possible topics.

## 2 Generative models for text

### 2.1 Latent Dirichlet Allocation (LDA) model

To simplify our discussion, we will use text modeling as a running example throughout this section, though it should be clear that the model is broadly applicable to general collections of discrete data.

In LDA, we assume that there are $k$ underlying latent topics according to which documents are generated, and that each topic is represented as a multinomial distribution over the $|V|$ words in the vocabulary. A document is generated by sampling a mixture of these topics and then sampling words from that mixture.

More precisely, a document of $N$ words $\mathbf{w} = \langle w_1, \ldots, w_N \rangle$ is generated by the following process. First, $\theta$ is sampled from a Dirichlet$(\alpha_1, \ldots, \alpha_k)$ distribution. This means that $\theta$ lies in the $(k-1)$-dimensional simplex: $\theta_i \geq 0, \sum_i \theta_i = 1$. Then, for each of the $N$ words, a topic $z_n \in \{1, \ldots, k\}$ is sampled from a Mult$(\theta)$ distribution $p(z_n = i|\theta) = \theta_i$. Finally, each word $w_n$ is sampled, conditioned on the $z_n$th topic, from the multinomial distribution $p(w|z_n)$. Intuitively, $\theta_i$ can be thought of as the degree to which topic $i$ is referred to in the document. Written out in full, the probability of a document is therefore the following mixture:

$$p(\mathbf{w}) = \int_\theta \left( \prod_{n=1}^{N} \sum_{z_n=1}^{k} p(w_n|z_n; \beta) p(z_n|\theta) \right) p(\theta; \alpha) d\theta, \tag{1}$$

where $p(\theta; \alpha)$ is Dirichlet, $p(z_n|\theta)$ is a multinomial parameterized by $\theta$, and $p(w_n|z_n; \beta)$ is a multinomial over the words. This model is parameterized by the $k$-dimensional Dirichlet parameters $\alpha = \langle \alpha_1, \ldots, \alpha_k \rangle$ and a $k \times |V|$ matrix $\beta$, which are parameters controlling the $k$ multinomial distributions over words. The graphical model representation of LDA is shown in Figure 1.

As Figure 1 makes clear, this model is *not* a simple Dirichlet-multinomial clustering model. In such a model the innermost plate would contain only $w_n$; the topic node would be sampled only once for each document; and the Dirichlet would be sampled only once for the whole collection. In LDA, the Dirichlet is sampled for each document, and the multinomial topic node is sampled *repeatedly* within the document. The Dirichlet is thus a component in the probability model rather than a prior distribution over the model parameters.

We see from Eq. (1) that there is a second interpretation of LDA. Having sampled $\theta$, words are drawn iid from the multinomial/unigram model given by $p(w|\theta) = \sum_{z=1}^{k} p(w|z) p(z|\theta)$. Thus, LDA is a mixture model where the unigram models $p(w|\theta)$ are the mixture components, and $p(\theta; \alpha)$ gives the mixture weights. Note that unlike a traditional mixture of unigrams model, this distribution has an infinite

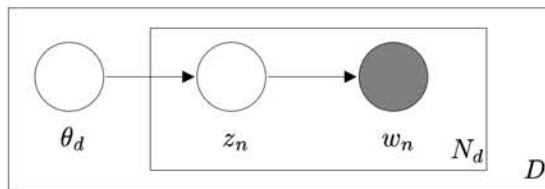

Figure 1: Graphical model representation of LDA. The boxes are plates representing replicates. The outer plate represents documents, while the inner plate represents the repeated choice of topics and words within a document.

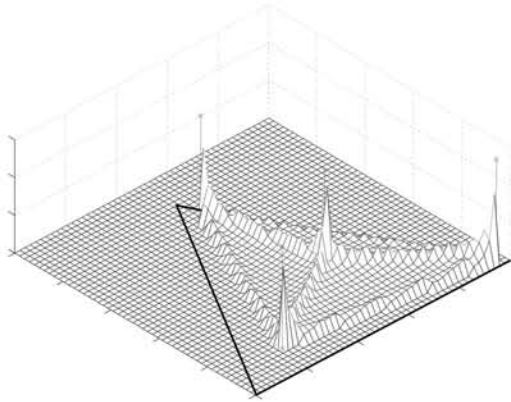

Figure 2: An example distribution on unigram models $p(w|\theta)$ under LDA for three words and four topics. The triangle embedded in the x-y plane is the 2-D simplex over all possible multinomial distributions over three words. (E.g., each of the vertices of the triangle corresponds to a deterministic distribution that assigns one of the words probability 1; the midpoint of an edge gives two of the words 0.5 probability each; and the centroid of the triangle is the uniform distribution over all 3 words). The four points marked with an **x** are the locations of the multinomial distributions $p(w|z)$ for each of the four topics, and the surface shown on top of the simplex is an example of a resulting density over multinomial distributions given by LDA.

number of continuously-varying mixture components indexed by $\theta$. The example in Figure 2 illustrates this interpretation of LDA as defining a random distribution over unigram models $p(w|\theta)$.

## 2.2   Related models

The mixture of unigrams model [6] posits that every document is generated by a *single* randomly chosen topic:

$$p(\mathbf{w}) = \sum_{z=1}^{k} \left( \prod_{n=1}^{N} p(w_n|z) \right) p(z). \tag{2}$$

This model allows for different documents to come from different topics, but fails to capture the possibility that a document may express multiple topics. LDA captures this possibility, and does so with an increase in the parameter count of only one parameter: rather than having $k-1$ free parameters for the multinomial $p(z)$ over the $k$ topics, we have $k$ free parameters for the Dirichlet.

A second related model is Hofmann's probabilistic latent semantic indexing (pLSI) [3], which posits that a document label $d$ and a word $w$ are conditionally independent given the hidden topic $z$:

$$p(d, w) = \sum_{z=1}^{k} p(w|z)p(z|d)p(d). \tag{3}$$

This model does capture the possibility that a document may contain multiple topics since $p(z|d)$ serve as the mixture weights of the topics. However, a subtlety of pLSI—and the crucial difference between it and LDA—is that $d$ is a dummy index into the list of documents in the training set. Thus, $d$ is a multinomial random variable with as many possible values as there are training documents, and the model learns

the topic mixtures $p(z|d)$ only for those documents on which it is trained. For this reason, pLSI is not a fully generative model and there is no clean way to use it to assign probability to a previously unseen document. Furthermore, the number of parameters in pLSI is on the order of $k|V| + k|D|$, where $|D|$ is the number of documents in the training set. Linear growth in the number of parameters with the size of the training set suggests that overfitting is likely to be a problem and indeed, in practice, a "tempering" heuristic is used to smooth the parameters of the model.

## 3 Inference and learning

Let us begin our description of inference and learning problems for LDA by examining the contribution to the likelihood made by a single document. To simplify our notation, let $w_n^j = 1$ iff $w_n$ is the $j$th word in the vocabulary and $z_n^i = 1$ iff $z_n$ is the $i$th topic. Let $\beta_{ij}$ denote $p(w^j = 1|z^i = 1)$, and $\mathbf{w} = \langle w_1, \dots, w_N \rangle$, $\mathbf{z} = \langle z_1, \dots, z_N \rangle$. Expanding Eq. (1), we have:

$$p(\mathbf{w}; \alpha, \beta) = \frac{\Gamma\left(\sum_i \alpha_i\right)}{\prod_i \Gamma(\alpha_i)} \int_\theta \left( \prod_{i=1}^k \theta_i^{\alpha_i - 1} \right) \left( \prod_{n=1}^N \sum_{i=1}^k \prod_{j=1}^{|V|} (\theta_i \beta_{ij})^{w_n^j} \right) d\theta \qquad (4)$$

This is a hypergeometric function that is infeasible to compute exactly [4].

Large text collections require fast inference and learning algorithms and thus we have utilized a variational approach [5] to approximate the likelihood in Eq. (4). We use the following variational approximation to the log likelihood:

$$
\begin{aligned}
\log p(\mathbf{w}; \alpha, \beta) &= \log \int_\theta \sum_{\mathbf{z}} p(\mathbf{w}|\mathbf{z}; \beta) p(\mathbf{z}|\theta) p(\theta; \alpha) \frac{q(\theta, \mathbf{z}; \gamma, \phi)}{q(\theta, \mathbf{z}; \gamma, \phi)} d\theta \\
&\geq \mathrm{E}_q[\log p(\mathbf{w}|\mathbf{z}; \beta) + \log p(\mathbf{z}|\theta) + \log p(\theta; \alpha) - \log q(\theta, \mathbf{z}; \gamma, \phi)],
\end{aligned}
$$

where we choose a fully factorized variational distribution $q(\theta, \mathbf{z}; \gamma, \phi) = q(\theta; \gamma) \prod_n q(z_n; \phi_n)$ parameterized by $\gamma$ and $\phi_n$, so that $q(\theta; \gamma)$ is Dirichlet$(\gamma)$, and $q(z_n; \phi_n)$ is Mult$(\phi_n)$. Under this distribution, the terms in the variational lower bound are computable and differentiable, and we can maximize the bound with respect to $\gamma$ and $\phi$ to obtain the best approximation to $p(\mathbf{w}; \alpha, \beta)$.

Note that the third and fourth terms in the variational bound are not straightforward to compute since they involve the entropy of a Dirichlet distribution, a $(k-1)$-dimensional integral over $\theta$ which is expensive to compute numerically. In the full version of this paper, we present a sequence of reductions on these terms which use the $\log \Gamma$ function and its derivatives. This allows us to compute the integral using well-known numerical routines.

Variational inference is coordinate ascent in the bound on the probability of a single document. In particular, we alternate between the following two equations until the objective converges:

$$\phi_{ni} \propto \beta_{iw_n} \exp\left( \Psi(\gamma_i) - \Psi\left( \textstyle\sum_{j=1}^k \gamma_j \right) \right), \qquad (5)$$

$$\gamma_i = \alpha_i + \textstyle\sum_{n=1}^N \phi_{ni} \qquad (6)$$

where $\Psi$ is the first derivative of the $\log \Gamma$ function. Note that the resulting variational parameters can also be used and interpreted as an approximation of the parameters of the true posterior.

In the current paper we focus on maximum likelihood methods for parameter estimation. Given a collection of documents $\mathcal{D} = \{\mathbf{w}_1, \dots, \mathbf{w}_M\}$, we utilize the EM

algorithm with a variational E step, maximizing a lower bound on the log likelihood:

$$\log p(\mathcal{D}) \geq \sum_{m=1}^{M} \mathrm{E}_{q_m}\left[\log p(\theta, \mathbf{z}, \mathbf{w})\right] - \mathrm{E}_{q_m}\left[\log q_m(\theta, \mathbf{z})\right]. \tag{7}$$

The E step refits $q_m$ for each document by running the inference step described above. The M step optimizes Eq. (7) with respect to the model parameters $\alpha$ and $\beta$. For the multinomial parameters $\beta_{ij}$ we have the following M step update equation:

$$\beta_{ij} \propto \sum_{m=1}^{M} \sum_{n=1}^{|\mathbf{w}_m|} \phi_{mni} w_{mn}^{j}. \tag{8}$$

The Dirichlet parameters $\alpha_i$ are not independent of each other and we apply Newton-Raphson to optimize them:

$$\frac{\partial \ell}{\partial \alpha_i} = \sum_{m=1}^{M} \left( \Psi\left(\sum_{j=1}^{k} \alpha_j\right) - \Psi(\alpha_i) \right) + \left( \Psi(\gamma_{mi}) - \Psi\left(\sum_{j=1}^{k} \gamma_{mj}\right) \right). \tag{9}$$

The variational EM algorithm alternates between maximizing Eq. (7) with respect to $q_m$ and with respect to $(\alpha, \beta)$ until convergence.

## 4 Experiments and Examples

We first tested LDA on two text corpora.[1] The first was drawn from the TREC AP corpus, and consisted of 2500 news articles, with a vocabulary size of $|V| = 37{,}871$ words. The second was the CRAN corpus, consisting of 1400 technical abstracts, with $|V| = 7747$ words.

We begin with an example showing how LDA can capture multiple-topic phenomena in documents. By examining the (variational) posterior distribution on the topic mixture $q(\theta; \gamma)$, we can identify the topics which were most likely to have contributed to many words in a given document; specifically, these are the topics $i$ with the largest $\gamma_i$. Examining the most likely words in the corresponding multinomials can then further tell us what these topics might be about. The following is an article from the TREC collection.

> The William Randolph Hearst Foundation will give \$1.25 million to Lincoln Center, Metropolitan Opera Co., New York Philharmonic and Juilliard School.
> "Our board felt that we had a real opportunity to make a mark on the future of the performing arts with these grants an act every bit as important as our traditional areas of support in health, medical research, education and the social services," Hearst Foundation President Randolph A. Hearst said Monday in announcing the grants.
> Lincoln Center's share will be \$200,000 for its new building, which will house young artists and provide new public facilities. The Metropolitan Opera Co. and New York Philharmonic will receive \$400,000 each. The Juilliard School, where music and the performing arts are taught, will get \$250,000.
> The Hearst Foundation, a leading supporter of the Lincoln Center Consolidated Corporate Fund, will make its usual annual \$100,000 donation, too.

Figure 3 shows the Dirichlet parameters of the corresponding variational distribution for those topics where $\gamma_i > 1$ ($k = 100$), and also lists the top 15 words (in

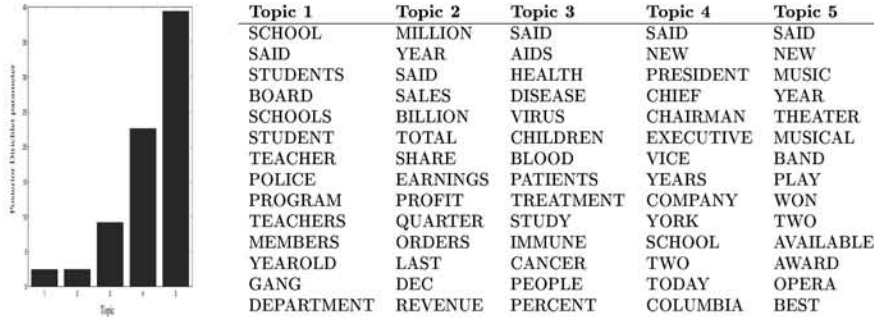

| Topic 1 | Topic 2 | Topic 3 | Topic 4 | Topic 5 |
|---------|---------|---------|---------|---------|
| SCHOOL | MILLION | SAID | SAID | SAID |
| SAID | YEAR | AIDS | NEW | NEW |
| STUDENTS | SAID | HEALTH | PRESIDENT | MUSIC |
| BOARD | SALES | DISEASE | CHIEF | YEAR |
| SCHOOLS | BILLION | VIRUS | CHAIRMAN | THEATER |
| STUDENT | TOTAL | CHILDREN | EXECUTIVE | MUSICAL |
| TEACHER | SHARE | BLOOD | VICE | BAND |
| POLICE | EARNINGS | PATIENTS | YEARS | PLAY |
| PROGRAM | PROFIT | TREATMENT | COMPANY | WON |
| TEACHERS | QUARTER | STUDY | YORK | TWO |
| MEMBERS | ORDERS | IMMUNE | SCHOOL | AVAILABLE |
| YEAROLD | LAST | CANCER | TWO | AWARD |
| GANG | DEC | PEOPLE | TODAY | OPERA |
| DEPARTMENT | REVENUE | PERCENT | COLUMBIA | BEST |

Figure 3: The Dirichlet parameters where $\gamma_i > 1$ ($k = 100$), and the top 15 words from the corresponding topics, for the document discussed in the text.

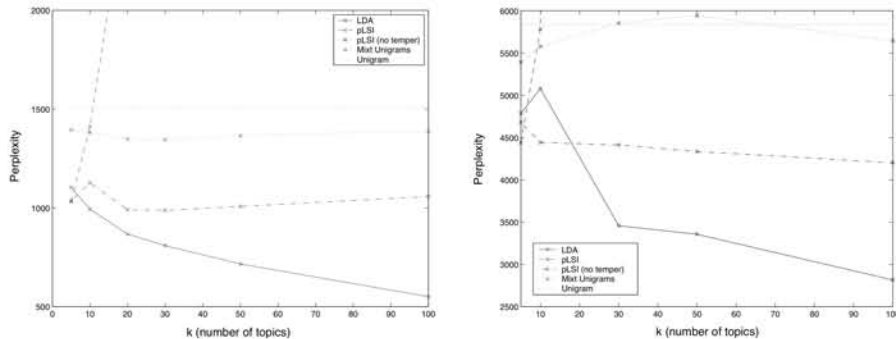

Figure 4: Perplexity results on the CRAN and AP corpora for LDA, pLSI, mixture of unigrams, and the unigram model.

order) from these topics. This document is mostly a combination of words about school policy (topic 4) and music (topic 5). The less prominent topics reflect other words about education (topic 1), finance (topic 2), and health (topic 3).

## 4.1 Formal evaluation: Perplexity

To compare the generalization performance of LDA with other models, we computed the perplexity of a test set for the AP and CRAN corpora. The perplexity, used by convention in language modeling, is monotonically decreasing in the likelihood of the test data, and can be thought of as the inverse of the per-word likelihood. More formally, for a test set of $M$ documents, perplexity($\mathcal{D}_{\text{test}}$) $= \exp\left(-\sum_m \log p(\mathbf{w}_m) / \sum_m |\mathbf{w}_m|\right)$.

We compared LDA to both the mixture of unigrams and pLSI described in Section 2.2. We trained the pLSI model with and without tempering to reduce overfitting. When tempering, we used part of the test set as the hold-out data, thereby giving it a slight unfair advantage. As mentioned previously, pLSI does not readily generate or assign probabilities to previously unseen documents; in our experiments, we assigned probability to a new document $d$ by marginalizing out the dummy training set indices[2]: $p(\mathbf{w}) = \sum_d (\prod_{n=1}^{N} \sum_z p(w_n|z)p(z|d))p(d)$.

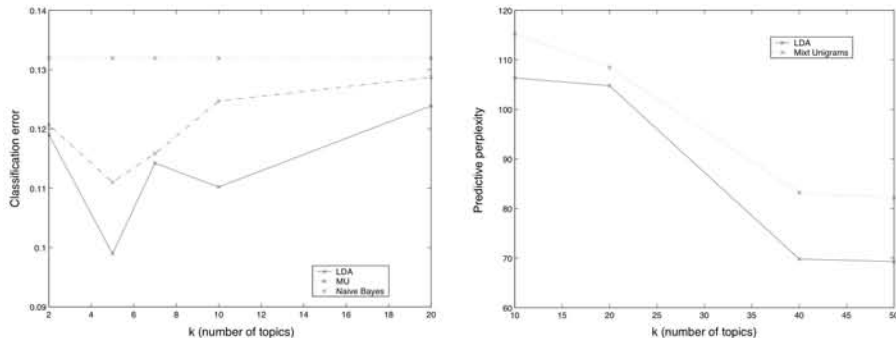

Figure 5: Results for classification (left) and collaborative filtering (right)

Figure 4 shows the perplexity for each model and both corpora for different values of $k$. The latent variable models generally do better than the simple unigram model. The pLSI model severely overfits when not tempered (the values beyond $k = 10$ are off the graph) but manages to outperform mixture of unigrams when tempered. LDA consistently does better than the other models. To our knowledge, these are by far the best text perplexity results obtained by a bag-of-words model.

## 4.2 Classification

We also tested LDA on a text classification task. For each class $c$, we learn a separate model $p(\mathbf{w}|c)$ of the documents in that class. An unseen document is classified by picking $\arg\max_c p(c|\mathbf{w}) = \arg\max_c p(\mathbf{w}|c)p(c)$. Note that using a simple unigram distribution for $p(\mathbf{w}|c)$ recovers the traditional naive Bayes classification model.

Using the same (standard) subset of the WebKB dataset as used in [6], we obtained classification error rates illustrated in Figure 5 (left). In all cases, the difference between LDA and the other algorithms' performance is statistically significant ($p < 0.05$).

## 4.3 Collaborative filtering

Our final experiment utilized the EachMovie collaborative filtering dataset. In this dataset a collection of users indicates their preferred movie choices. A user and the movies he chose are analogous to a document and the words in the document (respectively).

The collaborative filtering task is as follows. We train the model on a fully observed set of users. Then, for each test user, we are shown all but one of the movies that she liked and are asked to predict what the held-out movie is. The different algorithms are evaluated according to the likelihood they assign to the held-out movie. More precisely define the predictive perplexity on $M$ test users to be $\exp(-\sum_{m=1}^{M} \log p(w_{mN_d}|w_{m1}, \ldots, w_{m(N_d-1)})/M)$. With 5000 training users, 3500 testing users, and a vocabulary of 1600 movies, we find predictive perplexities illustrated in Figure 5 (right).

## 5 Conclusions

We have presented a generative probabilistic framework for modeling the topical structure of documents and other collections of discrete data. Topics are represented

explicitly via a multinomial variable $z_n$ that is repeatedly selected, once for each word, in a given document. In this sense, the model generates an allocation of the words in a document to topics. When computing the probability of a new document, this unknown allocation induces a mixture distribution across the words in the vocabulary. There is a many-to-many relationship between topics and words as well as a many-to-many relationship between documents and topics.

While Dirichlet distributions are often used as conjugate priors for multinomials in Bayesian modeling, it is preferable to instead think of the Dirichlet in our model as a component of the *likelihood*. The Dirichlet random variable $\theta$ is a latent variable that gives generative probabilistic semantics to the notion of a "document" in the sense that it allows us to put a distribution on the space of possible documents. The words that are actually obtained are viewed as a continuous mixture over this space, as well as being a discrete mixture over topics.[3]

The generative nature of LDA makes it easy to use as a module in more complex architectures and to extend it in various directions. We have already seen that collections of LDA can be used in a classification setting. If the classification variable is treated as a latent variable we obtain a mixture of LDA models, a useful model for situations in which documents cluster not only according to their topic overlap, but along other dimensions as well. Another extension arises from generalizing LDA to consider Dirichlet/multinomial mixtures of bigram or trigram models, rather than the simple unigram models that we have considered here. Finally, we can readily fuse LDA models which have different vocabularies (e.g., words and images); these models interact via a common abstract topic variable and can elegantly use both vocabularies in determining the topic mixture of a given document.

**Acknowledgments**

A. Ng is supported by a Microsoft Research fellowship. This work was also supported by a grant from Intel Corporation, NSF grant IIS-9988642, and ONR MURI N00014-00-1-0637.

## Footnotes

[1]To enable repeated large scale comparison of various models on large corpora, we implemented our variational inference algorithm on a parallel computing cluster. The (bottleneck) E step is distributed across nodes so that the $q_m$ for different documents are calculated in parallel.

[2]A second natural method, marginalizing out $d$ and $z$ to form a unigram model using the resulting $p(w)$'s, did not perform well (its performance was similar to the standard unigram model).

[3]These remarks also distinguish our model from the Bayesian Dirichlet/Multinomial allocation model (DMA)of [2], which is a finite alternative to the Dirichlet process. The DMA places a mixture of Dirichlet priors on $p(w|z)$ and sets $\alpha_i = \alpha_0$ for all $i$.

# References

[1] D. Cohn and T. Hofmann. The missing link—A probabilistic model of document content and hypertext connectivity. In *Advances in Neural Information Processing Systems 13*, 2001.

[2] P.J. Green and S. Richardson. Modelling heterogeneity with and without the Dirichlet process. *Technical Report, University of Bristol*, 1998.

[3] T. Hofmann. Probabilistic latent semantic indexing. *Proceedings of the Twenty-Second Annual International SIGIR Conference*, 1999.

[4] T. J. Jiang, J. B. Kadane, and J. M. Dickey. Computation of Carlson's multiple hypergeometric functions $r$ for Bayesian applications. *Journal of Computational and Graphical Statistics*, 1:231–251, 1992.

[5] M. I. Jordan, Z. Ghahramani, T. S. Jaakkola, and L. K. Saul. Introduction to variational methods for graphical models. *Machine Learning*, 37:183–233, 1999.

[6] K. Nigam, A. Mccallum, S. Thrun, and T. Mitchell. Text classification from labeled and unlabeled documents using EM. *Machine Learning*, 39(2/3):103–134, 2000.

[7] A. Popescul, L. H. Ungar, D. M. Pennock, and S. Lawrence. Probabilistic models for unified collaborative and content-based recommendation in sparse-data environments. In *Uncertainty in Artificial Intelligence, Proceedings of the Seventeenth Conference*, 2001.

